# Basis Construction from Power Series Expansions of Value Functions

**Sridhar Mahadevan**
Department of Computer Science
University of Massachusetts
Amherst, MA 01003
mahadeva@cs.umass.edu

**Bo Liu**
Department of Computer Science
University of Massachusetts
Amherst, MA 01003
boliu@cs.umass.edu

## Abstract

This paper explores links between basis construction methods in Markov decision processes and power series expansions of value functions. This perspective provides a useful framework to analyze properties of existing bases, as well as provides insight into constructing more effective bases. Krylov and Bellman error bases are based on the Neumann series expansion. These bases incur very large initial Bellman errors, and can converge rather slowly as the discount factor approaches unity. The Laurent series expansion, which relates discounted and average-reward formulations, provides both an explanation for this slow convergence as well as suggests a way to construct more efficient basis representations. The first two terms in the Laurent series represent the scaled average-reward and the average-adjusted sum of rewards, and subsequent terms expand the discounted value function using powers of a generalized inverse called the Drazin (or group inverse) of a singular matrix derived from the transition matrix. Experiments show that Drazin bases converge considerably more quickly than several other bases, particularly for large values of the discount factor. An incremental variant of Drazin bases called Bellman average-reward bases (BARBs) is described, which provides some of the same benefits at lower computational cost.

## 1 Introduction

Markov decision processes (MDPs) are a well-studied model of sequential decision-making under uncertainty [11]. Recently, there has been growing interest in automatic basis construction methods for constructing a problem-specific low-dimensional representation of an MDP. Functions on the original state space, such as the reward function or the value function, are "compressed" by projecting them onto a basis matrix $\Phi$, whose column space spans a low-dimensional subspace of the function space on the states of the original MDP. Among the various approaches proposed are *reward-sensitive* bases, such as Krylov bases [10] and an incremental variant called Bellman error basis functions (BEBFs) [9]. These approaches construct bases by *dilating* the (sampled) reward function by geometric powers of the (sampled) transition matrix of a policy. An alternative approach, called proto-value functions, constructs *reward-invariant* bases by finding the eigenvectors of the symmetric graph Laplacian matrix induced by the neighborhood topology of the state space under the given actions [7].

A fundamental dilemma that is revealed by these prior studies is that neither reward-sensitive nor reward-invariant eigenvector bases by themselves appear to be fully satisfactory. A Chebyshev polynomial bound for the error due to approximation using Krylov bases was derived in [10], extending a known similar result for general Krylov approximation [12]. This bound shows that performance of Krylov bases (and BEBFs) tends to degrade as the discount factor $\gamma \to 1$. Intuitively, the initial basis vectors capture short-term transient behavior near rewarding regions, and tend to poorly ap-

proximate the value function over the entire state space until a sufficiently large time scale is reached. A straightforward geometrical analysis of approximation errors using least-squares fixed point approximation onto a basis shows that the Bellman error decomposes into the sum of two terms: a reward error and a second term involving the feature prediction error [1, 8] (see Figure 1). This analysis helps reveal sources of error: Krylov bases and BEBFs tend to have low reward error (or zero in the non-sampled case), and hence a large component of the error in using these bases tends to be due to the feature prediction error. In contrast, PVFs tend to have large reward error since typical spiky goal reward functions are poorly approximated by smooth low-order eigenvectors; however, their feature prediction error can be quite low as the eigenvectors often capture invariant subspaces of the model transition matrix.

A hybrid approach that combined low-order eigenvectors of the transition matrix (or PVFs) with higher-order Krylov bases was proposed in [10], which empirically resulted in a better approach. This paper demonstrates a more principled approach to address this problem, by constructing new bases that emerge from investigating the links between basis construction methods and different power series expansions of value functions. In particular, instead of using the eigenvectors of the transition matrix, the proposed approach uses the average-reward or gain as the first basis vector, and dilates the reward function by powers of the average-adjusted transition matrix. It turns out that the gain is an element of the space of eigenvectors associated with the eigenvalue $\lambda = 1$ of the transition matrix. The relevance of power series expansion to approximations of value functions was hinted at in early work by Schwartz [13] on undiscounted optimization, although he did not discuss basis construction.

Krylov and Bellman error basis functions (BEBFs) [10, 9, 12], as well as proto-value functions [7], can be related to terms in the Neumann series expansion. Ultimately, the performance of these bases is limited by the speed of convergence of the Neumann expansion, and of course, other errors arising due to reward and feature prediction error. The key insight underlying this paper is to exploit connections between average-reward and discounted formulations. It is well-known that discounted value functions can be written in the form of a Laurent series expansion, where the first two terms correspond to the average-reward term (scaled by $\frac{1}{1-\gamma}$), and the average-adjusted sum of rewards (or *bias*). Higher order terms involve powers of the Drazin (or group) inverse of a singular matrix related to the transition matrix. This expansion provides a mathematical framework for analyzing the properties of basis construction methods and developing newer representations. In particular, Krylov bases converge slowly for high discount factors since the value function is dominated by the scaled average-reward term, which is poorly approximated by the initial BEBF or Krylov basis vectors as it involves the long-term limiting matrix $P^*$. The Laurent series expansion leads to a new type of basis called a Drazin basis [6]. An approximation of Drazin bases called Bellman average-reward bases (BARBs) is described and compared with BEBFs, Krylov bases, and PVFs.

## 2 MDPs and Their Approximation

A Markov decision process $M$ is formally defined by the tuple $(S, A, P, R)$, where $S$ is a discrete state space, $A$ is the set of actions (which could be conditioned on the state $s$, so that $A_s$ is the set of legal actions in $s$), $P(s'|s, a)$ is the transition matrix specifying the effect of executing action $a$ in state $s$, and $R(s, a) : S \times A \to \mathbb{R}$ is the (expected) reward for doing action $a$ in state $s$. The value function $V$ associated with a deterministic policy $\pi : S \to A$ is defined as the long-term expected sum of rewards received starting from a state, and following the policy $\pi$ indefinitely. [1] The value function $V$ associated with a fixed policy $\pi$ can be determined by solving the Bellman equation

$$V = T(V) = R + \gamma PV,$$

where $T(.)$ is the Bellman backup operator, $R(s) = R(s, \pi(s))$, $P(s, s') = P(s'|s, \pi(s))$, and the discount factor $0 \leq \gamma < 1$. For a fixed policy $\pi$, the induced discounted Markov reward process is defined as $(P, R, \gamma)$.

A popular approach to approximating $V$ is to use a linear combination of basis functions $V \approx \hat{V} = \Phi w$, where the basis matrix $\Phi$ is of size $|S| \times k$, and $k \ll |S|$. The *Bellman error* for a given basis $\Phi$, denoted $BE(\Phi)$, is defined as the difference between the two sides of the Bellman equation, when

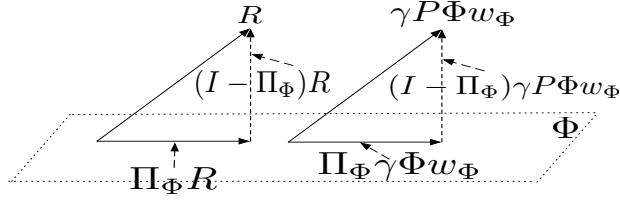

Figure 1: The Bellman error due to a basis and its decomposition. See text for explanation.

$V$ is approximated by $\Phi w$. As Figure 1 illustrates, simple geometrical considerations can be used to show that the Bellman error can be decomposed into two terms: a *reward error* term and a *weighted feature error* term [1, 8]: [2]

$$BE(\Phi) = R + \gamma P \Phi w_\Phi - \Phi w_\Phi = (I - \Pi_\Phi)R + (I - \Pi_\Phi)\gamma P \Phi w_\Phi,$$

where $\Pi_\Phi$ is the weighted orthogonal projector onto the column space spanned by the basis $\Phi$. $w_\Phi$ is the weight vector associated with the fixed point $\Phi w_\phi = \Pi_\Phi(T(\Phi w_\Phi))$. If the Markov chain defined by $P$ is irreducible and aperiodic, then $\Pi_\Phi = \Phi(\Phi^T D^* \Phi)^{-1}\Phi^T D^*$, where $D^*$ is a diagonal matrix whose entries contain the stationary distribution of the Markov chain. In the experiments shown below, for simplicity we will use the unweighted projection $\Pi_\Phi = \Phi(\Phi^T \Phi)^{-1}\Phi^T$, in which case the fixed point is given by $w_\phi = (\Phi^T \Phi - \gamma \Phi^T P \Phi)^{-1}\Phi^T R$.

## 3 Neumann Expansions and Krylov/BEBF Bases

The most familiar expansion of the value function $V$ is in terms of the Neumann series, where

$$V = (I - \gamma P)^{-1} R = (I + \gamma P + \gamma^2 P^2 + \ldots)R.$$

Krylov bases correspond to successive terms in the Neumann series [10, 12]. The $j^{th}$ *Krylov* subspace $\mathcal{K}_j$ is defined as the space spanned by the vectors: $\mathcal{K}_j = \{R, PR, P^2 R, \ldots, P^{j-1}R\}$. Note that $\mathcal{K}_1 \subseteq \mathcal{K}_2 \subseteq \ldots$, such that for some $m, \mathcal{K}_m = \mathcal{K}_{m+1} = \mathcal{K}$ (where $m$ is the minimal polynomial of $A = I - \gamma P$). Thus, $\mathcal{K}$ is the $P$-invariant Krylov space generated by $P$ and $R$. An incremental variant of the Krylov-based approach is called Bellman error basis functions (BEBFs) [9]. In particular, given a set of basis functions $\Phi_k$ (where the first one is assumed to equal $R$), the next basis is defined to be $\phi_{k+1} = T(\Phi_k w_{\Phi_k}) - \Phi_k w_{\Phi_k}$. In the model-free reinforcement learning setting, $\phi_{k+1}$ can be approximated by the temporal-difference (TD) error $\phi_{k+1} = r + \gamma \hat{Q}_k(s', \pi_k(s')) - \hat{Q}_k(s, a)$, given a set of stored transitions in the form $(s, a, r, s')$. Here, $\hat{Q}_k$ is the fixed-point least-squares approximation to the action-value function $Q(s, a)$ on the basis $\Phi_k$. It can be easily shown that BEBFs and Krylov bases define the same space [8].

### 3.1 Convergence Analysis

A key issue in evaluating the effectiveness of Krylov bases (and BEBFs) is the speed of convergence of the Neumann series. As $\gamma \to 1$, Krylov bases and BEBFs converge rather slowly, owing to a large increase in the weighted feature error. In practice, this problem can be shown to be acute even for values of $\gamma = 0.9$ or $\gamma = 0.99$, which are quite common in experiments. Petrik [10] derived a bound for the error due to Krylov approximation, which depends on the condition number of $I - \gamma P$, and the ratio of two Chebyshev polynomials on the complex plane. The condition number of $I - \gamma P$ can significantly increase as $\gamma \to 1$ (see Figure 2).

It has been shown that BEBFs reduce the Bellman error at a rate bounded by value iteration [9]. Iterative solution methods for solving linear systems $Ax = b$ can broadly be categorized as different ways of decomposing $A = S - T$, giving rise to the iteration $Sx_{k+1} = Tx_k + b$. The convergence of this iteration depends on the spectral structure of $B = S^{-1}T$, in particular its largest eigenvalue. For standard value iteration, $A = I - \gamma P$, and consequently the natural decomposition is to set

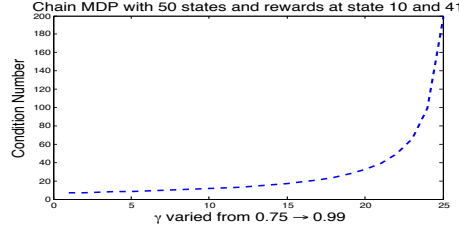

Figure 2: Condition number of $I - \gamma P$ as $\gamma \to 1$, where $P$ is the transition matrix of the optimal policy in a chain MDP of 50 states with rewards in states 10 and 41 [9].

$S = I$ and $T = \gamma P$. Thus, the largest eigenvalue of $S^{-1}T$ is $\gamma$, and as $\gamma \to 1$, convergence of value iteration is progressively decelerated. For Krylov bases, the following bound can be shown.

**Theorem 1:** The Bellman error in approximating the value function for a discounted Markov reward process $(P, R, \gamma)$ using $m$ BEBF or Krylov bases is is bounded by

$$\|BE(\Phi)\|_2 = \|(I - \Pi_\Phi)\gamma P \Phi w_\Phi\|_2 \le \kappa(X) \frac{C_m(\frac{a}{d})}{C_m(\frac{c}{d})},$$

where the (Jordan form) diagonalization of $I - \gamma P = XSX^{-1}$, and $\kappa(X) = \|X\|_2\|X^{-1}\|_2$ is the condition number of $I - \gamma P$. $C_m$ is the Chebyshev polynomial of degree $m$ of the first kind, where $a, c$ and $d$ are chosen such that $E(a, c, d)$ is an ellipse on the set of complex numbers that covers all the eigenvalues of $I - \gamma P$ with center $c$, focal distance $d$, and major semi-axis $a$.

**Proof:** This result follows directly from standard Krylov space approximation results [12], and past results on approximation using BEBFs and Krylov spaces for MDPs [10, 8]. First, note that the overall Bellman error can be reduced to the weighted feature error, since the reward error is 0 as $R$ is in the span of both BEBFs and Krylov bases:

$$\|BE(\Phi)\|_2 = \|T(\Phi w_\Phi) - \Phi w_\Phi\|_2 = \|R - (I - \gamma P)\hat{V}\|_2 = \|(I - \Pi_\Phi)\gamma P \Phi w_\Phi\|_2.$$

Next, setting $A = (I - \gamma P)$, we have

$$\|R - Aw\|_2 = \|R - \sum_{i=1}^{m} w_i A^i R\|_2 = \|\sum_{i=0}^{m} -w(i) A^i R\|_2.$$

assuming $w(0) = -1$. A standard result in Krylov space approximation [12] shows that

$$\min_{p \in \mathcal{P}_m} \|p(A)\|_2 \le \min_{p \in \mathcal{P}_m} \kappa(X) \max_{i=1,\dots,n} |p(\lambda_i)| \le \kappa(X) \frac{C_m(\frac{a}{d})}{C_m(\frac{c}{d})},$$

where $\mathcal{P}_m$ is the set of polynomials of degree $m$. $\qquad\square$

Figure 2 shows empirically that one reason for the slow convergence of BEBFs and Krylov bases is that as $\gamma \to 1$, the condition number of $I - \gamma P$ significantly increases. Figure 3 compares the weighted feature error of BEBF bases (the performance of Krylov bases is identical and not shown) on a 50 state chain domain with a single goal reward of 1 in state 25. The dynamics of the chain are identical to those in [9]. Notice as $\gamma$ increases, the feature error increases dramatically.

## 4 Laurent Series Expansion and Drazin Bases

A potential solution to the slow convergence of BEBFs and Krylov bases is suggested by a different power series called the Laurent expansion. It is well known from the classical theory of MDPs that the discounted value function can be written in a form that relates it to the average-reward formulation [11]. This connection uses the following Laurent series expansion of $V$ in terms of the average reward $\rho$ of the policy $\pi$, the average-adjusted sum of rewards $h$, and higher order terms that involve the generalized spectral inverse (Drazin or group inverse) of $I - P$.

$$V = \frac{1}{\gamma}\left(\frac{\gamma}{1-\gamma}\rho + h + \sum_{n=1}^{\infty}(\frac{\gamma-1}{\gamma})^n((I-P)^D)^{n+1}R\right). \tag{1}$$

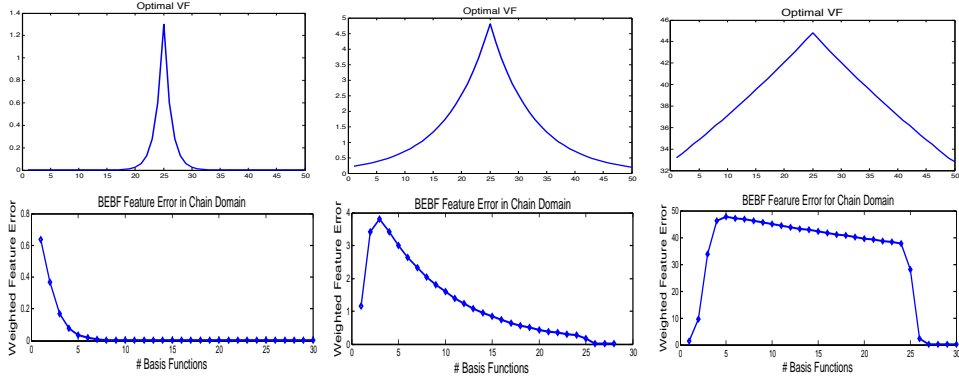

Figure 3: Weighted feature error of BEBF bases on a 50 state chain MDP with a single reward of 1 in state 25. Top: optimal value function for $\gamma = 0.5, 0.9, 0.99$. Bottom: Weighted feature error. As $\gamma$ increases, the weighted feature error grows much larger as the value function becomes progressively smoother and less like the reward function. Note the difference in scale between the three plots.

As $\gamma \rightarrow 1$, the first term in the Laurent series expansion grows quite large causing the slow convergence of Krylov bases and BEBFs. $(I - P)^D$ is a generalized inverse of the singular matrix $I - P$ called the *Drazin* inverse [2, 11]. For any square matrix $A \in \mathbb{C}^{n \times n}$, the Drazin inverse $X$ of $A$ satisfies the following properties: (1) $XAX = X$ (2) $XA = AX$ (3) $A^{k+1}X = A^k$. Here, $k$ is the *index* of matrix $A$, which is the smallest nonnegative integer $k$ such that $\mathcal{R}(A^k) = \mathcal{R}(A^{k+1})$. $\mathcal{R}(A)$ is the range (or column space) of matrix $A$. For example, a nonsingular (square) matrix $A$ has index 0, because $\mathcal{R}(A^0) = \mathcal{R}(I) = \mathcal{R}(A)$. The matrix $I - P$ of a Markov chain has index $k = 1$. For index 1 matrices, the Drazin inverse turns out to be the same as the *group inverse*, which is defined as

$$(I - P)^D = (I - P + P^*)^{-1} - P^*,$$

where the long-term limiting matrix $P^* = \lim_{n \to \infty} \frac{1}{n} \sum_{k=0}^{n} P^k = I - (I - P)(I - P)^D$. The matrix $(I - P + P^*)^{-1}$ is often referred to as the *fundamental matrix* of a Markov chain. Note that for index 1 matrices, the Drazin (or group) inverse satisfies the additional property $AXA = A$. Also, $P^*$ and $I - P^*$ are orthogonal projection matrices, since they are both idempotent and furthermore $PP^* = P^*P = P^*$, and $P^*(I - P^*) = 0$. [3] The gain and bias can be expressed in terms of $P^*$. In particular, the gain $g = P^*R$, and the bias or average-adjusted value function is given by:

$$h = (I - P)^D R = \left[ (I - P + P^*)^{-1} - P^* \right] R = \sum_{t=0}^{\infty} (P^t - P^*) R.$$

where the last equality holds for aperiodic Markov chains. If we represent the coefficients in the Laurent series as $y_{-1}, y_0, \ldots,$, they can be shown to be solutions to the following set of equations (for $n = 1, 2, \ldots$). In terms of the expansion above, $y_{-1}$ is the gain of the policy, $y_0$ is its bias, and so on.

$$(I - P)y_{-1} = 0, \;\; y_{-1} + (I - P)y_0 = R, \; \ldots \;, y_{n-1} + (I - P)y_n = 0.$$

Analogous to the Krylov bases, the successive terms of the Laurent series expansion can be viewed as basis vectors. More formally, the *Drazin basis* is defined as the space spanned by the vectors [6]:

$$D_m = \{P^*R, (I - P)^D R, ((I - P)^D)^2 R, \ldots, ((I - P)^D)^{m-1} R\}. \tag{2}$$

The first basis vector is the average-reward or gain $g = P^*R$ of policy $\pi$. The second basis vector is the bias, or average-adjusted sum of rewards $h$. Subsequent basis vectors correspond to higher-order terms in the Laurent series.

# 5 Bellman Average-Reward Bases

To get further insight into methods for approximating Drazin bases, it is helpful to note that the $(i,j)^{th}$ element in the Drazin or group inverse matrix is the difference between the expected number of visits to state $j$ starting in state $i$ following the transition matrix $P$ versus the expected number of visits to $j$ following the long-term limiting matrix $P^*$. Building on this insight, an approximate Drazin basis called Bellman average-reward bases (BARBs) can be defined as follows. First, the approximate Drazin basis is defined as the space spanned by the vectors

$$
\begin{aligned}
A_m &= \{P^*R, (P-P^*)R, (P-P^*)^2 R, \dots, (P-P^*)^{m-1}R\} \\
&= \{P^*R, PR - P^*R, P^2 R - P^*R, \dots, P^{m-1}R - P^*R\}.
\end{aligned}
$$

BARBs are similar to Krylov bases, except that the reward function is being dilated by the *average-adjusted* transition matrix $P - P^*$, and the first basis element is the gain. Defining $\rho = P^*R$, BARBs can be defined as follows:

$$
\begin{aligned}
\phi_1 &= \rho = P^*R. \\
\phi_{k+1} &= R - \rho + P\Phi_k w_{\Phi_k} - \Phi_k w_{\Phi_k}.
\end{aligned}
$$

The cost of computing BARBs is essentially that of computing BEBFs (or Krylov bases), except for the term involving the gain $\rho$. Analogous to BEBFs, in the model-free reinforcement learning setting, BARBs can be computed using the average-adjusted TD error

$$
\phi_{k+1}(s) = r - \rho_k(s) + \hat{Q}_k(s', \pi_k(s')) - \hat{Q}_k(s, a).
$$

There are a number of incremental algorithms for computing $\rho$ (such as the scheme used in $R$-learning [13], or simply averaging the sample rewards). Several methods for computing $P^*$ are discussed in [14].

## 5.1 Expressivity Properties

Some results concerning the expressivity of approximate Drazin bases and BARBs are now discussed. Due to space, detailed proofs are not included.

**Theorem 2** For any $k > 1$, the following hold:

$$
\begin{aligned}
span\left\{A_k(R)\right\} &\subseteq span\left\{BARB_{k+1}(R)\right\}. \\
span\left\{BARB_{k+1}(R)\right\} &= span\left\{\{R\} \cup A_k(R)\right\}. \\
span\left\{BEBF_k(R)\right\} &\subseteq span\left\{BARB_{k+1}(R)\right\}. \\
span\left\{BARB_{k+1}(R)\right\} &= span\left\{\{\rho\} \cup BEBF_k(R)\right\}.
\end{aligned}
$$

**Proof:** Proof follows by induction. For $k = 1$, both approximate Drazin bases and BARBs contain the gain $\rho$. For $k = 2$, $BARB_2(R) = R - \rho$, whereas $A_2 = PR - \rho$ (which is included in $BARB_3(R)$). For general $k > 2$, the new basis vector in $A_k$ is $P^{k-1}R$, which can be shown to be part of $BARB_{k+1}(R)$. The other results can be derived through similar analysis. $\square$

There is a similar decomposition of the average-adjusted Bellman error into a component that depends on the average-adjusted reward error and an undiscounted weighted feature error.

**Theorem 3** Given a basis $\Phi$, for any average reward Markov reward process $(P, R)$, the Bellman error can be decomposed as follows:

$$
\begin{aligned}
T(\hat{V}) - \hat{V} &= R - \rho + P\Phi w_\Phi - \Phi w_\Phi \\
&= (I - \Pi_\Phi)(R - \rho) + (I - \Pi_\Phi)P\Phi w_\Phi \\
&= (I - \Pi_\Phi)R - (I - \Pi_\Phi)\rho + (I - \Pi_\Phi)P\Phi w_\Phi.
\end{aligned}
$$

**Proof:** The three terms represent the reward error, the average-reward error, and the undiscounted weighted feature error. The proof follows immediately from the geometry of the Bellman error, similar to that shown in Figure 1, and using the property of linearity of orthogonal projectors. $\square$

A more detailed convergence analysis of BARBs is given in [4], based on the relationship between the approximation error and the mixing rate of the Markov chain defined by $P$.

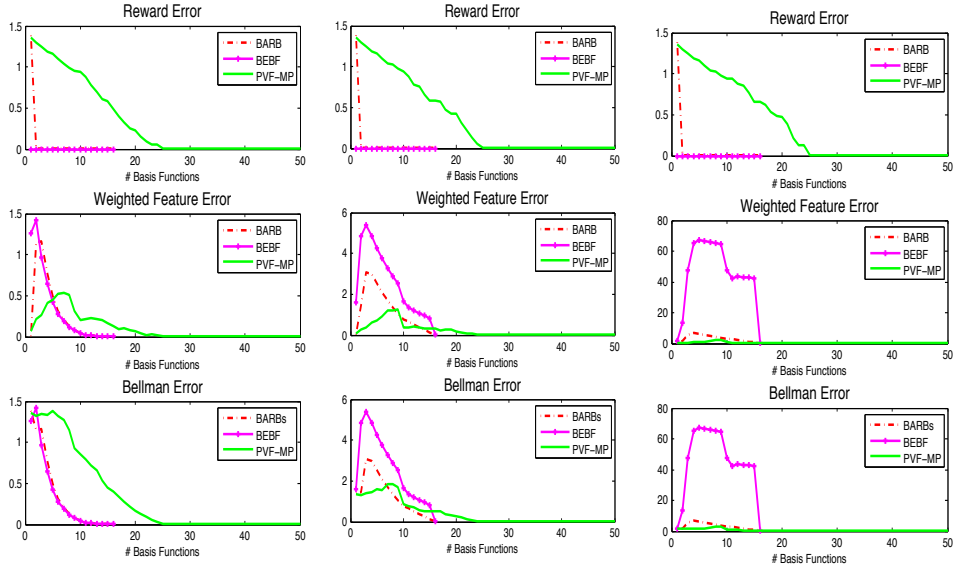

Figure 4: Experimental comparison on a 50 state chain MDP with rewards in state 10 and 41. Left column: $\gamma = 0.7$. Middle column: $\gamma = 0.9$. Right column: $\gamma = 0.99$.

## 6 Experimental Comparisons

Figure 4 compares the performance of Bellman average-reward basis functions (BARBs) vs. Bellman-error basis functions (BEBFs), and a variant of proto-value functions (PVF-MP) on a 50 state chain MDP. This problem was previously studied by [3]. The two actions (go left, or go right) succeed with probability 0.9. When the actions fail, they result in movement in the opposite direction with probability 0.1. The two ends of the chain are treated as "dead ends". Rewards of $+1$ are given in states 10 and 41. The PVF-MP algorithm selects basis functions incrementally based upon the Bellman error, where basis function k+1 is the PVF that has the largest inner product with the Bellman error resulting from the previous k basis functions.

PVFs have a high reward error, since the reward function is a set of two delta functions that is poorly approximated by the eigenvectors of the combinatorial Laplacian on the chain graph. However, PVFs have very low weighted feature error. The overall Bellman error remains large due to the high reward error. The reward error for BEBFs is by definition 0 as $R$ is a basis vector itself. However, the weighted feature error for BEBFs grows quite large as $\gamma$ increases from 0.7 to 0.99, particularly initially, until around 15 bases are used. Consequently, the Bellman error for BEBFs remains large initially. BARBs have the best overall performance at this task, particularly for $\gamma = 0.9$ and 0.99.

The plots in Figure 5 compare BARBs vs. Drazin and Krylov bases in the two-room gridworld MDP [7]. Drazin bases perform the best, followed by BARBs, and then Krylov bases. At higher discount factors, the differences are more noticeable. Finally, Figure 6 compares BARBs vs. BEBFs on a $10 \times 10$ grid world MDP with a reward placed in the upper left corner state. The advantage of using BARBs over BEBFs is significant as $\gamma \to 1$. The policy is a random walk on the grid. Finally, similar results were also obtained in experiments conducted on random MDPs, where the states were decomposed into communicating classes of different block sizes (not shown).

## 7 Conclusions and Future Work

The Neumann and Laurent series lead to different ways of constructing problem-specific bases. The Neumann series, which underlies Bellman error and Krylov bases, tends to converge slowly as $\gamma \to 1$. To address this shortcoming, the Laurent series was used to derive a new approach called the Drazin basis, which expands the discounted value function in terms of the average-reward, the bias, and higher order terms representing powers of the Drazin inverse of a singular matrix derived from

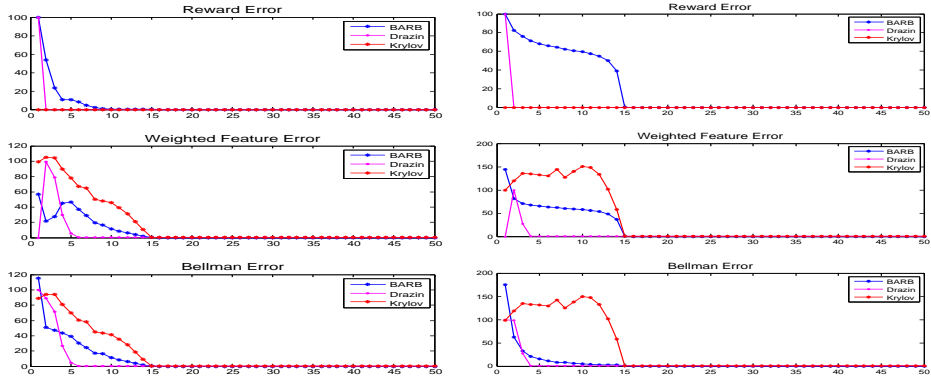

Figure 5: Comparison of BARBs vs. Drazin and Krylov bases in a 100 state two-room MDP [7]. All bases were evaluated on the optimal policy. The reward was set at $+100$ for reaching a corner goal state in one of the rooms. Left: $\gamma = 0.9$. Right: $\gamma = 0.99$.

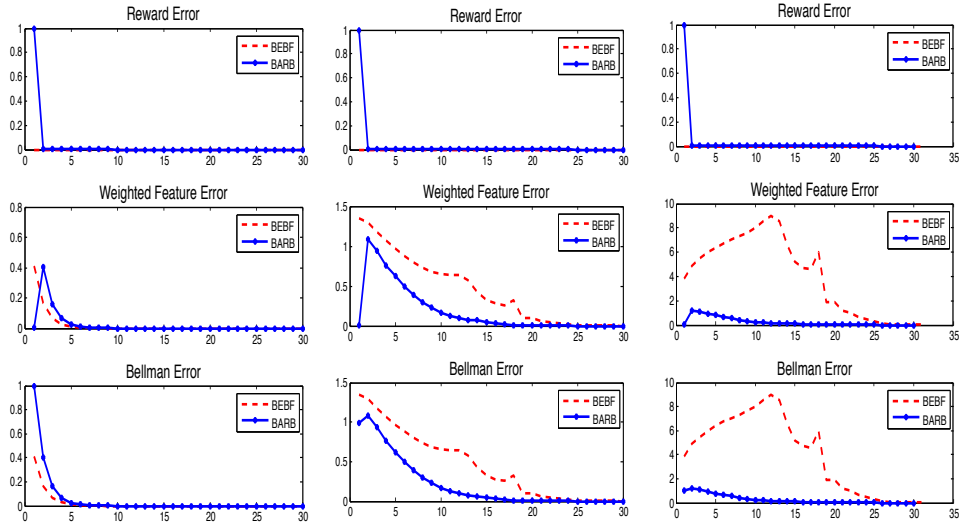

Figure 6: Experimental comparison of BARBs and BEBFs on a $10 \times 10$ grid world MDP with a reward in the upper left corner. Left: $\gamma = 0.7$. Middle: $\gamma = 0.99$. Right: $\gamma = 0.999$.

the transition matrix. An incremental version of Drazin bases called Bellman average-reward bases (BARBs) was investigated. Numerical experiments on simple MDPs show superior performance of Drazin bases and BARBs to BEBFs, Krylov bases, and PVFs. Scaling BARBs and Drazin bases to large MDPs requires addressing sampling issues, and exploiting structure in transition matrices, such as using factored representations. BARBs are computationally more tractable than Drazin bases, and merit further study. Reinforcement learning methods to estimate the first few terms of the Laurent series were proposed in [5], and can be adapted for basis construction. The Schultz expansion provides a way of rewriting the Neumann series using a multiplicative series of dyadic powers of the transition matrix, which is useful for multiscale bases [6].

## Acknowledgements

This research was supported in part by the National Science Foundation (NSF) under grants NSF IIS-0534999 and NSF IIS-0803288, and Air Force Office of Scientific Research (AFOSR) under grant FA9550-10-1-0383. Any opinions, findings, and conclusions or recommendations expressed in this material are those of the authors and do not necessarily reflect the views of the AFOSR or the NSF.

## Footnotes

[1]In what follows, we suppress the dependence of $P$, $R$, and $V$ on the policy $\pi$ to avoid clutter.

[2]The two components of the Bellman error may partially (or fully) cancel each other out: the Bellman error of $V$ itself is 0, but it generates non-zero reward and feature prediction errors.

[3]Several methods are available to compute Drazin inverses, as described in [2]. An iterative method called *Successive Matrix Squaring* (SMS) has also has been developed for efficient parallel implementation [15].

# References

[1] D. Bertsekas and D. Castañon. Adaptive aggregation methods for infinite horizon dynamic programming. *IEEE Transactions on Automatic Control*, 34:589–598, 1989.

[2] S. Campbell and C. Meyer. *Generalized Inverses of Linear Transformations*. Pitman, 1979.

[3] M. Lagoudakis and R. Parr. Least-squares policy iteration. *Journal of Machine Learning Research*, 4:1107–1149, 2003.

[4] B. Liu and S. Mahadevan. An investigation of basis construction from power series expansions of value functions. Technical report, University Massachusetts, Amherst, 2010.

[5] S Mahadevan. Sensitive-discount optimality: Unifying discounted and average reward reinforcement learning. In *Proceedings of the International Conference on Machine Learning*, 1996.

[6] S. Mahadevan. Learning representation and control in Markov Decision Processes: New frontiers. *Foundations and Trends in Machine Learning*, 1(4):403–565, 2009.

[7] S. Mahadevan and M. Maggioni. Proto-value functions: A Laplacian framework for learning representation and control in Markov Decision Processes. *Journal of Machine Learning Research*, 8:2169–2231, 2007.

[8] R. Parr, , Li. L., G. Taylor, C. Painter-Wakefield, and M. Littman. An analysis of linear models, linear value-function approximation, and feature selection for reinforcement learning. In *Proceedings of the International Conference on Machine Learning (ICML)*, 2008.

[9] R. Parr, C. Painter-Wakefield, L. Li, and M. Littman. Analyzing feature generation for value function approximation. In *Proceedings of the International Conference on Machine Learning (ICML)*, pages 737–744, 2007.

[10] M. Petrik. An analysis of Laplacian methods for value function approximation in MDPs. In *Proceedings of the International Joint Conference on Artificial Intelligence (IJCAI)*, pages 2574–2579, 2007.

[11] M. L. Puterman. *Markov Decision Processes*. Wiley Interscience, New York, USA, 1994.

[12] Y. Saad. *Iterative Methods for Sparse Linear Systems*. SIAM Press, 2003.

[13] A. Schwartz. A reinforcement learning method for maximizing undiscounted rewards. In *Proc. 10th International Conf. on Machine Learning*. Morgan Kaufmann, San Francisco, CA, 1993.

[14] William J. Stewart. Numerical methods for computing stationary distributions of finite irreducible markov chains. In *Advances in Computational Probability*. Kluwer Academic Publishers, 1997.

[15] Y. Wei. Successive matrix squaring algorithm for computing the Drazin inverse. *Applied Mathematics and Computation*, 108:67–75, 2000.

